# Mixture Density Estimation

**Jonathan Q. Li**
Department of Statistics
Yale University
P.O. Box 208290
New Haven, CT 06520
*Qiang.Li@aya.yale.edu*

**Andrew R. Barron**
Department of Statistics
Yale University
P.O. Box 208290
New Haven, CT 06520
*Andrew.Barron@yale.edu*

## Abstract

Gaussian mixtures (or so-called radial basis function networks) for density estimation provide a natural counterpart to sigmoidal neural networks for function fitting and approximation. In both cases, it is possible to give simple expressions for the iterative improvement of performance as components of the network are introduced one at a time. In particular, for mixture density estimation we show that a k-component mixture estimated by maximum likelihood (or by an iterative likelihood improvement that we introduce) achieves log-likelihood within order $1/k$ of the log-likelihood achievable by any convex combination. Consequences for approximation and estimation using Kullback-Leibler risk are also given. A Minimum Description Length principle selects the optimal number of components $k$ that minimizes the risk bound.

## 1 Introduction

In density estimation, Gaussian mixtures provide flexible-basis representations for densities that can be used to model heterogeneous data in high dimensions. We introduce an index of regularity $c_f$ of density functions $f$ with respect to mixtures of densities from a given family. Mixture models with $k$ components are shown to achieve Kullback-Leibler approximation error bounded by $c_f^2/k$ for every $k$. Thus in a manner analogous to the treatment of sinusoidal and sigmoidal networks in Barron [1],[2], we find classes of density functions $f$ such that reasonable size networks (not exponentially large as function of the input dimension) achieve suitable approximation and estimation error.

Consider a parametric family $G = \{\phi_\theta(x), x \in \mathcal{X} \subset R^{d'} : \theta \in \Theta \subset R^d\}$ of probability density functions parameterized by $\theta \in \Theta$. Then consider the class $\mathcal{C} = CONV(G)$ of density functions for which there is a mixture representation of the form

$$f_P(x) = \int_\Theta \phi_\theta(x) P(d\theta) \tag{1}$$

where $\phi_\theta(x)$ are density functions from $G$ and $P$ is a probability measure on $\Theta$.

The main theme of the paper is to give approximation and estimation bounds of arbitrary densities by finite mixture densities. We focus our attention on densities

inside $C$ first and give an approximation error bound by finite mixtures for arbitrary $f \in C$. The approximation error is measured by Kullback-Leibler divergence between two densities, defined as

$$D(f\|g) = \int f(x) \log[f(x)/g(x)]dx. \qquad (2)$$

In density estimation, $D$ is more natural to use than the $L^2$ distance often seen in the function fitting literature. Indeed, $D$ is invariant under scale transformations (and other 1-1 transformation of the variables) and it has an intrinsic connection with Maximum Likelihood, one of the most useful methods in the mixture density estimation. The following result quantifies the approximation error.

**THEOREM 1** *Let $G = \{\phi_\theta(x) : \theta \in \Theta\}$ and $C = CONV(G)$. Let $f(x) = \int \phi_\theta(x)P(d\theta) \in C$. There exists $f_k$, a $k$-component mixture of $\phi_\theta$, such that*

$$D(f\|f_k) \le \frac{c_f^2 \gamma}{k}. \qquad (3)$$

In the bound, we have

$$c_f^2 = \int \frac{\int \phi_\theta^2(x)P(d\theta)}{\int \phi_\theta(x)P(d\theta)} dx, \qquad (4)$$

and $\gamma = 4[\log(3\sqrt{e}) + a]$, where

$$a = \sup_{\theta_1,\theta_2,x} \log \frac{\phi_{\theta_1}(x)}{\phi_{\theta_2}(x)}. \qquad (5)$$

Here, $a$ characterizes an upper bound of the log ratio of the densities in $G$, when the parameters are restricted to $\Theta$ and the variable to $\mathcal{X}$.

Note that the rate of convergence, $1/k$, is not related to the dimensions of $\Theta$ or $\mathcal{X}$. The behavior of the constants, though, depends on the choices of $G$ and the target $f$.

For example we may take $G$ to be the Gaussian location family, which we restrict to a set $\mathcal{X}$ which is a cube of side-length $A$. Likewise we restrict the parameters to be in the same cube. Then,

$$a \le \frac{dA^2}{\sigma^2}. \qquad (6)$$

In this case, $a$ is linear in dimension.

The value of $c_f^2$ depends on the target density $f$. Suppose $f$ is a finite mixture with $M$ components, then

$$c_f^2 \le M, \qquad (7)$$

with equality if and only if those $M$ components are disjoint. Indeed, suppose $f(x) = \sum_{i=1}^{M} p_i \phi_{\theta_i}(x)$, then $p_i \phi_{\theta_i}(x) / \sum_{i=1}^{M} p_i \phi_{\theta_i}(x) \le 1$ and hence

$$c_f^2 = \int \frac{\sum_{i=1}^{M}(p_i\phi_{\theta_i}(x))\phi_{\theta_i}(x)}{\sum_{i=1}^{M} p_i\phi_{\theta_i}(x)} dx \le \int \sum_{i=1}^{M}(1)\phi_{\theta_i}(x)dx = M. \qquad (8)$$

Genovese and Wasserman [3] deal with a similar setting. A Kullback-Leibler approximation bound of order $1/\sqrt{k}$ for one-dimensional mixtures of Gaussians is given by them.

In the more general case that $f$ is not necessarily in $C$, we have a competitive optimality result. Our density approximation is nearly at least as good as any $g_P$ in $C$.

**THEOREM 2** *For every $g_P(x) = \int \phi_\theta(x) P(d\theta)$,*

$$D(f\|f_k) \leq D(f\|g_P) + \frac{c_{f,P}^2}{k}\gamma. \tag{9}$$

*Here,*

$$c_{f,P}^2 = \int \frac{\int \phi_\theta^2(x) P(d\theta)}{(\int \phi_\theta(x) P(d\theta))^2} f(x) dx. \tag{10}$$

In particular, we can take infimum over all $g_P \in \mathcal{C}$, and still obtain a bound.

Let $D(f\|\mathcal{C}) = \inf_{g \in \mathcal{C}} D(f\|g)$. A theory of information projection shows that if there exists a sequence of $f_k$ such that $D(f\|f_k) \to D(f\|\mathcal{C})$, then $f_k$ converges to a function $f^*$, which achieves $D(f\|\mathcal{C})$. Note that $f^*$ is not necessarily an element in $\mathcal{C}$. This is developed in Li[4] building on the work of Bell and Cover[5]. As a consequence of Theorem 2 we have

$$D(f\|f_k) \leq D(f\|f^*) + \frac{c_{f,*}^2}{k}\gamma \tag{11}$$

where $c_{f,*}^2$ is the smallest limit of $c_{f,P}^2$ for sequences of $P$ achieving $D(f\|g_P)$ that approaches the infimum $D(f\|\mathcal{C})$.

We prove Theorem 1 by induction in the following section. An appealing feature of such an approach is that it provides an iterative estimation procedure which allows us to estimate one component at a time. This greedy procedure is shown to perform almost as well as the full-mixture procedures, while the computational task of estimating one component is considerably easier than estimating the full mixtures.

Section 2 gives the iterative construction of a suitable approximation, while Section 3 shows how such mixtures may be estimated from data. Risk bounds are stated in Section 4.

## 2   An iterative construction of the approximation

We provide an iterative construction of $f_k$'s in the following fashion. Suppose during our discussion of approximation that $f$ is given. We seek a $k$-component mixture $f_k$ close to $f$. Initialize $f_1$ by choosing a single component from $G$ to minimize $D(f\|f_1) = D(f\|\phi_\theta)$. Now suppose we have $f_{k-1}(x)$. Then let $f_k(x) = (1-\alpha)f_{k-1}(x) + \alpha\phi_\theta(x)$ where $\alpha$ and $\theta$ are chosen to minimize $D(f\|f_k)$. More generally let $f_k$ be any sequence of $k$-component mixtures, for $k = 1, 2, \ldots$ such that $D(f\|f_k) \leq \min_{\alpha,\theta} D(f\|(1-\alpha)f_{k-1} + \alpha\phi_\theta)$. We prove that such sequences $f_k$ achieve the error bounds in Theorem 1 and Theorem 2.

Those familiar with the iterative Hilbert space approximation results of Jones[6], Barron[1], and Lee, Bartlett and Williamson[7], will see that we follow a similar strategy. The use of $L_2$ distance measures for density approximation involves $L_2$ norms of component densities that are exponentially large with dimension. Naive Taylor expansion of the Kullback-Leibler divergence leads to an $L_2$ norm approximation (weighted by the reciprocal of the density) for which the difficulty remains (Zeevi & Meir[8], Li[9]). The challenge for us was to adapt iterative approximation to the use of Kullback-Leibler divergence in a manner that permits the constant $a$ in the bound to involve the *logarithm* of the density ratio (rather than the ratio itself) to allow more manageable constants.

The proof establishes the inductive relationship

$$D_k \le (1 - \alpha)D_{k-1} + \alpha^2 B, \tag{12}$$

where $B$ is bounded and $D_k = D(f\|f_k)$. By choosing $\alpha_1 = 1, \alpha_2 = 1/2$ and thereafter $\alpha_k = 2/k$, it's easy to see by induction that $D_k \le 4B/k$.

To get (12), we establish a quadratic upper bound for $-\log \frac{f_k}{f} = -\log \frac{((1-\alpha)f_{k-1} + \alpha\phi_\theta)}{f}$. Three key analytic inequalities regarding to the logarithm will be handy for us,

$$-\log(r) \le -(r - 1) + [\frac{-\log(r_0) + r_0 - 1}{(r_0 - 1)^2}](r - 1)^2 \tag{13}$$

for $r \ge r_0 > 0$,

$$2[\frac{-\log(r) + r - 1}{r - 1}] \le \log r, \tag{14}$$

and

$$\frac{-\log(r) + r - 1}{(r - 1)^2} \le 1/2 + \log^-(r) \tag{15}$$

where $\log^-(\cdot)$ is the negative part of the logarithm. The proof of of inequality (13) is done by verifying that $\frac{-\log(r)+r-1}{(r-1)^2}$ is monotone decreasing in $r$. Inequalities (14) and (15) are shown by separately considering the cases that $r < 1$ and $r > 1$ (as well as the limit as $r \to 1$). To get the inequalities one multiplies through by $(r-1)$ or $(r-1)^2$, respectively, and then takes derivatives to obtain suitable monotonicity in $r$ as one moves away from $r = 1$.

Now apply the inequality (13) with $r = \frac{(1-\alpha)f_{k-1}+\alpha\phi_\theta}{g}$ and $r_0 = \frac{(1-\alpha)f_{k-1}}{g}$, where $g$ is an arbitrary density in $C$ with $g = \int \phi_\theta P(d\theta)$. Note that $r \ge r_0$ in this case because $\frac{\alpha\phi_\theta}{g} \ge 0$. Plug in $r = r_0 + \alpha\frac{\phi_\theta}{g}$ at the right side of (13) and expand the square. Then we get

$$
\begin{aligned}
-\log(r) &\le -(r_0 + \frac{\alpha\phi}{g} - 1) + [\frac{-\log(r_0) + r_0 - 1}{(r_0 - 1)^2}][(r_0 - 1) + (\frac{\alpha\phi}{g})]^2 \\
&= -\frac{\alpha\phi}{g} - \log(r_0) + \alpha^2\frac{\phi^2}{g^2}[\frac{-\log(r_0) + r_0 - 1}{(r_0 - 1)^2}] + 2\alpha\frac{\phi}{g}[\frac{-\log(r_0) + r_0 - 1}{r_0 - 1}].
\end{aligned}
$$

Now apply (14) and (15) respectively. We get

$$-\log(r) \le -\log(r_0) - \frac{\alpha\phi}{g} + \alpha^2\frac{\phi^2}{g^2}(1/2 + \log^-(r_0)) + \alpha\frac{\phi}{g}\log(r_0). \tag{16}$$

Note that in our application, $r_0$ is a ratio of densities in $C$. Thus we obtain an upper bound for $\log^-(r_0)$ involving $a$. Indeed we find that $(1/2 + \log^-(r_0)) \le \gamma/4$ where $\gamma$ is as defined in the theorem.

In the case that $f$ is in $C$, we take $g = f$. Then taking the expectation with respect to $f$ of both sides of (16), we acquire a quadratic upper bound for $D_k$, noting that $r = \frac{f_k}{f}$. Also note that $D_k$ is a function of $\theta$. The greedy algorithm chooses $\theta$ to minimize $D_k(\theta)$. Therefore

$$D_k \le \min_\theta D_k(\theta) \le \int D_k(\theta)P(d\theta). \tag{17}$$

Plugging the upper bound (16) for $D_k(\theta)$ into (17), we have

$$D_k \le \int_\theta \int_x [-\log(r_0) - \frac{\alpha\phi}{g} + \alpha^2\frac{\phi^2}{g^2}(\gamma/4) + \alpha\frac{\phi}{g}\log(r_0)]f(x)dx P(d\theta). \tag{18}$$

where $r_0 = (1 - \alpha)f_{k-1}(x)/g(x)$ and $P$ is chosen to satisfy $\int_\theta \phi_\theta(x)P(d\theta) = g(x)$. Thus

$$D_k \leq (1 - \alpha)D_{k-1} + \alpha^2 \int \frac{\phi_\theta^2(x)P(d\theta)}{(g(x))^2} f(x)dx(\gamma/4) + \alpha \log(1 - \alpha) - \alpha - \log(1 - \alpha). \tag{19}$$

It can be shown that $\alpha \log(1 - \alpha) - \alpha - \log(1 - \alpha) \leq 0$. Thus we have the desired inductive relationship,

$$D_k \leq (1 - \alpha)D_{k-1} + \alpha^2 c_{f,P}^2 \gamma/4. \tag{20}$$

Therefore, $D_k \leq \frac{\gamma c_{f,P}^2}{k}$.

In the case that $f$ does not have a mixture representation of the form $\int \phi_\theta P(d\theta)$, i.e. $f$ is outside the convex hull $\mathcal{C}$, we take $D_k$ to be $\int f(x) \log \frac{g_P(x)}{f_k(x)} dx$ for any given $g_P(x) = \int \phi_\theta(x)P(d\theta)$. The above analysis then yields $D_k = D(f\|f_k) - D(f\|g_P) \leq \frac{\gamma c_{f,P}^2}{k}$ as desired. That completes the proof of Theorems 1 and 2.

## 3  A greedy estimation procedure

The connection between the K-L divergence and the MLE helps to motivate the following estimation procedure for $f_k$ if we have data $X_1, ..., X_n$ sampled from $f$. The iterative construction of $f_k$ can be turned into a sequential maximum likelihood estimation by changing $\min D(f\|f_k)$ to $\max \sum_{i=1}^n \log f_k(X_i)$ at each step. A surprising result is that the resulting estimator $\hat{f}_k$ has a log likelihood almost at least as high as log likelihood achieved by any density $g_P$ in $\mathcal{C}$ with a difference of order $1/k$. We formally state it as

$$\frac{1}{n}\sum_{i=1}^n \log \hat{f}_k(X_i) \geq \frac{1}{n}\sum_{i=1}^n \log g_P(X_i) - \gamma \frac{c_{F_n,P}^2}{k} \tag{21}$$

for all $g_P \in \mathcal{C}$. Here $F_n$ is the empirical distribution, for which $c_{F_n,P}^2 = (1/n)\sum_{i=1}^n c_{X_i,P}^2$ where

$$c_{x,P}^2 = \frac{\int \phi_\theta^2(x)P(d\theta)}{(\int \phi_\theta(x)P(d\theta))^2}. \tag{22}$$

The proof of this result (21) follows as in the proof in the last section, except that now we take $D_k = E_{F_n} \log g_P(X)/f_k(X)$ to be the expectation with respect to $F_n$ instead of with respect to the density $f$.

Let's look at the computation at each step to see the benefits this new greedy procedure can bring for us. We have $\hat{f}_k(x) = (1 - \alpha)\hat{f}_{k-1}(x) + \alpha\phi_\theta(x)$ with $\theta$ and $\alpha$ chosen to maximize

$$\sum_{i=1}^n \log[(1 - \alpha)\hat{f}_{k-1}(X_i) + \alpha\phi_\theta(X_i)] \tag{23}$$

which is a simple two component mixture problem, with one of the two components, $\hat{f}_{k-1}(x)$, fixed. To achieve the bound in (21), $\alpha$ can either be chosen by this iterative maximum likelihood or it can be held fixed at each step to equal $\alpha_k$ (which as before is $\alpha_k = 2/k$ for $k > 2$). Thus one may replace the MLE-computation of a $k$-component mixture by successive MLE-computations of two-component mixtures. The resulting estimate is guaranteed to have almost at least as high a likelihood as is achieved by any mixture density.

A disadvantage of the greedy procedure is that it may take a number of steps to adequately downweight poor initial choices. Thus it is advisable at each step to re-tune the weights of convex combinations of previous components (and even perhaps to adjust the locations of these components), in which case, the result from the previous iterations (with $k - 1$ components) provide natural initialization for the search at step $k$. The good news is that as long as for each $k$, given $\hat{f}_{k-1}$, the $\hat{f}_k$ is chosen among $k$ component mixtures to achieve likelihood at least as large as the choice achieving $\max_\theta \sum_{i=1}^n \log[(1 - \alpha_k)\hat{f}_{k-1}(X_i) + \alpha_k\phi_\theta(X_i)]$, that is, we require that

$$\sum_{i=1}^n \log \hat{f}_k(X_i) \geq \max_\theta \sum_{i=1}^n \log[(1 - \alpha_k)\hat{f}_{k-1}(X_i) + \alpha_k\phi_\theta(X_i)], \qquad (24)$$

then the conclusion (21) will follow.

In particular, our likelihood results and risk bound results apply both to the case that $\hat{f}_k$ is taken to be global maximizer of the likelihood over $k$-component mixtures as well as to the case that $\hat{f}_k$ is the result of the greedy procedure.

## 4   Risk bounds for the MLE and the iterative MLE

The metric entropy of the family $G$ is controlled to obtain the risk bound and to determine the precisions with which the coordinates of the parameter space are allowed to be represented. Specifically, the following Lipschitz condition is assumed: for $\theta \in \Theta \subset R^d$ and $x \in \mathcal{X} \subset R^d$,

$$\sup_{x \in \mathcal{X}} |\log \phi_\theta(x) - \log \phi_{\theta'}(x)| \leq B \sum_{j=1}^d |\theta_j - \theta'_j| \qquad (25)$$

where $\theta_j$ is the j-th coordinate of the parameter vector. Note that such a condition is satisfied by a Gaussian family with $x$ restricted to a cube with sidelength $A$ and has a location parameter $\theta$ that is also prescribed to be in the same cube. In particular, if we let the variance be $\sigma^2$, we may set $B = 2A/\sigma^2$.

Now we can state the bound on the K-L risk of $\hat{f}_k$.

**THEOREM 3** *Assume the condition (25). Also assume $\Theta$ to be a cube with side-length $A$. Let $\hat{f}_k(x)$ be either the maximizer of the likelihood over $k$-component mixtures or more generally any sequence of density estimates $\hat{f}_k$ satisfying (24). We have*

$$E(D(f\|\hat{f}_k)) - D(f\|\mathcal{C}) \leq \gamma^2 \frac{c_{f,*}^2}{k} + \gamma \frac{2kd}{n} \log(nABe). \qquad (26)$$

From the bound on risk, a best choice of $k$ would be of order roughly $\sqrt{n}$ leading to a bound on $ED(f\|\hat{f}_k) - D(f\|\mathcal{C})$ of order $1/\sqrt{n}$ to within logarithmic factors. However the best such bound occurs with $k = \gamma c_{f,*} \sqrt{n}/\sqrt{2d\log(nABe)}$ which is not available when the value of $c_{f,*}$ is unknown. More importantly, $k$ should not be chosen merely to optimize an upper bound on risk, but rather to balance whatever approximation and estimation sources of error actually occur. Toward this end we optimize a penalized likelihood criterion related to the minimum description length principle, following Barron and Cover [10].

Let $l(k)$ be a function of $k$ that satisfies $\sum_{k=1}^\infty e^{-l(k)} \leq 1$, such as $l(k) = 2\log(k+1)$.

A penalized MLE (or MDL) procedure picks $k$ by minimizing

$$\frac{1}{n} \sum_{i=1}^{n} \log \frac{1}{\hat{f}_k(X_i)} + 2kd\frac{\log(nABe)}{n} + 2l(k)/n. \tag{27}$$

Then we have

$$E(D(f\|\hat{f}_{\hat{k}})) - D(f\|\mathcal{C}) \leq \min_k \{\gamma^2 \frac{c_{f,*}^2}{k} + \gamma \frac{2kd}{n} \log(nABe) + 2l(k)/n\}. \tag{28}$$

A proof of these risk bounds is given in Li[4]. It builds on general results for maximum likelihood and penalized maximum likelihood procedures.

Recently, Dasgupta [11] has established a randomized algorithm for estimating mixtures of Gaussians, in the case that data are drawn from a finite mixture of sufficiently separated Gaussian components with common covariance, that runs in time linear in the dimension and quadratic in the sample size. However, present forms of his algorithm require impractically large sample sizes to get reasonably accurate estimates of the density. It is not yet known how his techniques will work for more general mixtures. Here we see that iterative likelihood maximization provides a better relationship between accuracy, sample size and number of components.

# References

[1] Barron, Andrew (1993) Universal Approximation Bounds for Superpositions of a Sigmoidal Function. *IEEE Transactions on Information Theory* **39**, No. 3: 930-945

[2] Barron, Andrew (1994) Approximation and Estimation Bounds for Artificial Neural Networks. *Machine Learning* **14**: 115-133.

[3] Genovese, Chris and Wasserman, Larry (1998) Rates of Convergence for the Gaussian Mixture Seive. Manuscript.

[4] Li, Jonathan Q. (1999) Estimation of Mixture Models. Ph.D Dissertation. The Department of Statistics. Yale University.

[5] Bell, Robert and Cover, Thomas (1988) Game-theoretic optimal portfolios. *Management Science* **34**: 724-733.

[6] Jones, Lee (1992) A simple lemma on greedy approximation in Hilbert space and convergence rates for projection pursuit regression and neural network training. *Annals of Statistics* **20**: 608-613.

[7] Lee, W.S., Bartlett, P.L. and Williamson R.C. (1996) Efficient Agnostic Learning of Neural Networks with Bounded Fan-in. *IEEE Transactions on Information Theory* **42**, No. 6: 2118-2132.

[8] Zeevi, Assaf and Meir Ronny (1997) Density Estimation Through Convex Combinations of Densities: Approximation and Estimation Bounds. *Neural Networks* **10**, No.1: 99-109.

[9] Li, Jonathan Q. (1997) Iterative Estimation of Mixture Models. Ph.D. Prospectus. The Department of Statistics. Yale University.

[10] Barron, Andrew and Cover, Thomas (1991) Minimum Complexity Density Estimation. *IEEE Transactions on Information Theory* **37**: 1034-1054.

[11] Dasgupta, Sanjoy (1999) Learning Mixtures of Gaussians. *Proc. IEEE Conf. on Foundations of Computer Science*, 634-644.